# Clustering Sequences with Hidden Markov Models

**Padhraic Smyth**
Information and Computer Science
University of California, Irvine
CA 92697-3425
smyth@ics.uci.edu

## Abstract

This paper discusses a probabilistic model-based approach to clustering *sequences*, using hidden Markov models (HMMs). The problem can be framed as a generalization of the standard mixture model approach to clustering in feature space. Two primary issues are addressed. First, a novel parameter initialization procedure is proposed, and second, the more difficult problem of determining the number of clusters $K$, from the data, is investigated. Experimental results indicate that the proposed techniques are useful for revealing hidden cluster structure in data sets of sequences.

## 1  Introduction

Consider a data set $D$ consisting of $N$ sequences, $D = \{S_1, \ldots, S_N\}$. $S_i = (\underline{x}_1^i, \ldots \underline{x}_{L_i}^i)$ is a sequence of length $L_i$ composed of potentially multivariate feature vectors $\underline{x}$. The problem addressed in this paper is the discovery from data of a natural grouping of the sequences into $K$ clusters. This is analogous to clustering in multivariate feature space which is normally handled by methods such as $k$-means and Gaussian mixtures. Here, however, one is trying to cluster the sequences $S$ rather than the feature vectors $\underline{x}$. As an example Figure 1 shows four sequences which were generated by two different models (hidden Markov models in this case). The first and third came from a model with "slower" dynamics than the second and fourth (details will be provided later). The sequence clustering problem consists of being given sample sequences such as those in Figure 1 and inferring from the data what the underlying clusters are. This is non-trivial since the sequences can be of different lengths and it is not clear what a meaningful distance metric is for sequence comparison.

The use of hidden Markov models for clustering sequences appears to have first

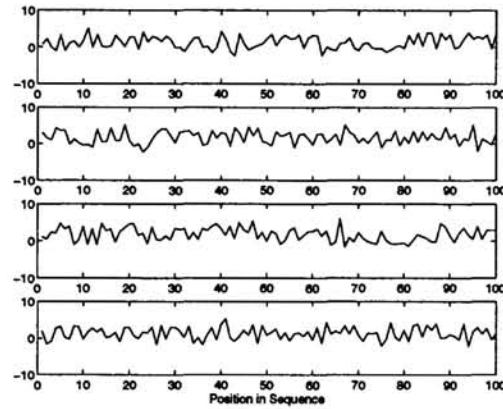

Figure 1: Which sequences came from which hidden Markov model ?

been mentioned in Juang and Rabiner (1985) and subsequently used in the context of discovering subfamilies of protein sequences in Krogh et al. (1994). This present paper contains two new contributions in this context: a cluster-based method for initializing the model parameters and a novel method based on cross-validated likelihood for determining automatically how many clusters to fit to the data.

A natural probabilistic model for this problem is that of a finite mixture model:

$$f_K(S) = \sum_{j=1}^{K} f_j(S|\theta_j)p_j \qquad (1)$$

where $S$ denotes a sequence, $p_j$ is the weight of the $j$th model, and $f_j(S|\theta_j)$ is the density function for the sequence data $S$ given the component model $f_j$ with parameters $\theta_j$. Here we will assume that the $f_j$'s are HMMs: thus, the $\theta_j$'s are the transition matrices, observation density parameters, and initial state probabilities, all for the $j$th component. $f_j(S|\theta_j)$ can be computed via the forward part of the forward backward procedure. More generally, the component models could be any probabilistic model for $S$ such as linear autoregressive models, graphical models, non-linear networks with probabilistic semantics, and so forth.

It is important to note that the motivation for this problem comes from the goal of building a *descriptive* model for the data, rather than *prediction* per se. For the prediction problem there is a clearly defined metric for performance, namely average prediction error on out-of-sample data (cf. Rabiner et al. (1989) in a speech context with clusters of HMMs and Zeevi, Meir, and Adler (1997) in a general time-series context). In contrast, for descriptive modeling it is not always clear what the appropriate metric for evaluation is, particularly when $K$, the number of clusters, is unknown. In this paper a density estimation viewpoint is taken and the likelihood of out-of-sample data is used as the measure of the quality of a particular model.

## 2   An Algorithm for Clustering Sequences into $K$ Clusters

Assume first that $K$, the number of clusters, is known. Our model is that of a mixture of HMMs as in Equation 1. We can immediately observe that this mixture can itself be viewed as a single "composite" HMM where the transition matrix $A$ of the model is block-diagonal, e.g., if the mixture model consists of two components with transition matrices $A_1$ and $A_2$ we can represent the overall mixture model as

a single HMM (in effect, a hierarchical mixture) with transition matrix

$$A = \begin{pmatrix} A_1 & 0 \\ 0 & A_2 \end{pmatrix} \qquad (2)$$

where the initial state probabilities are chosen appropriately to reflect the relative weights of the mixture components (the $p_k$ in Equation 1). Intuitively, a sequence is generated from this model by initially randomly choosing either the "upper" matrix $A_1$ (with probability $p_1$) or the "lower" matrix with probability $A_2$ (with probability $1 - p_1$) and then generating data according to the appropriate $A_i$. There is no "crossover" in this mixture model: data are assumed to come from one component or the other. Given this composite HMM a natural approach is to try to learn the parameters of the model using standard HMM estimation techniques, i.e., some form of initialization followed by Baum-Welch to maximize the likelihood. Note that unlike predictive modelling (where likelihood is not necessarily an appropriate metric to evaluate model quality), likelihood maximization is exactly what we want to do here since we seek a generative (descriptive) model for the data. We will assume throughout that the *number of states per component* is known a priori, i.e., that we are looking for $K$ HMM components each of which has $m$ states and $m$ is known. An obvious extension is to address the problem of learning $K$ and $m$ simultaneously but this is not dealt with here.

## 2.1  Initialization using Clustering in "Log-Likelihood Space"

Since the EM algorithm is effectively hill-climbing the likelihood surface, the quality of the final solution can depend critically on the initial conditions. Thus, using as much prior information as possible about the problem to seed the initialization is potentially worthwhile. This motivates the following scheme for initializing the $A$ matrix of the composite HMM:

1. Fit $N$ $m$-state HMMs, one to *each* individual sequence $S_i, 1 \leq i \leq N$. These HMMs can be initialized in a "default" manner: set the transition matrices uniformly and set the means and covariances using the $k$-means algorithm, where here $k = m$, not to be confused with $K$, the number of HMM components. For discrete observation alphabets modify accordingly.

2. For each fitted model $M_i$, evaluate the log-likelihood of each of the $N$ sequences given model $M_i$, i.e., calculate $L_{ij} = \log L(S_j|M_i), 1 \leq i, j \leq N$.

3. Use the log-likelihood distance matrix to cluster the sequences into $K$ groups (details of the clustering are discussed below).

4. Having pooled the sequences into $K$ groups, fit $K$ HMMs, one to each group, using the default initialization described above. From the $K$ HMMs we get $K$ sets of parameters: initialize the composite HMM in the obvious way, i.e., the $m \times m$ "block-diagonal" component $A_j$ of $A$ (where $A$ is $mK \times mK$) is set to the estimated transition matrix from the $j$th group and the means and covariances of the $j$th set of states are set accordingly. Initialize the $p_j$ in Equation 1 to $N_j/N$ where $N_j$ is the number of sequences which belong to cluster $j$.

After this initialization step is complete, learning proceeds directly on the composite HMM (with matrix $A$) in the usual Baum-Welch fashion using all of the sequences. The intuition behind this initialization procedure is as follows. The hypothesis is that the data are being generated by $K$ models. Thus, if we fit models to each individual sequence, we will get noisier estimates of the model parameters (than if we used all of the sequences from that cluster) but the parameters should be

clustered in some manner into $K$ groups about their true values (assuming the model is correct). Clustering directly in parameter space would be inappropriate (how does one define distance?): however, the log-likelihoods are a natural way to define pairwise distances.

Note that step 1 above requires the training of $N$ sequences individually and step 2 requires the evaluation of $N^2$ distances. For large $N$ this may be impractical. Suitable modifications which train only on a small random sample of the $N$ sequences and randomly sample the distance matrix could help reduce the computational burden, but this is not pursued here. A variety of possible clustering methods can be used in step 3 above. The "symmetrized distance" $L_{ij} = 1/2(L(S_i|M_j)+L(S_j|M_i))$ can be shown to be an appropriate measure of dissimilarity between models $M_i$ and $M_j$ (Juang and Rabiner 1985). For the results described in this paper, hierarchical clustering was used to generate $K$ clusters from the symmetrized distance matrix. The "furthest-neighbor" merging heuristic was used to encourage compact clusters and worked well empirically, although there is no particular reason to use only this method.

We will refer to the above clustering-based initialization followed by Baum-Welch training on the composite model as the "HMM-Clustering" algorithm in the rest of the paper.

## 2.2  Experimental Results

Consider a deceptively simple "toy" problem. 1-dimensional feature data are generated from a 2-component HMM mixture ($K = 2$), each with 2 states. We have

$$A_1 = \begin{pmatrix} 0.6 & 0.4 \\ 0.4 & 0.6 \end{pmatrix} \qquad A_2 = \begin{pmatrix} 0.4 & 0.6 \\ 0.6 & 0.4 \end{pmatrix}$$

and the observable feature data obey a Gaussian density in each state with $\sigma_1 = \sigma_2 = 1$ for each state in each component, and $\mu_1 = 0, \mu_2 = 3$ for the respective mean of each state of each component. 4 sample sequences are shown in Figure 1. The top, and third from top, sequences are from the "slower" component $A_1$ (is more likely to stay in any state than switch). In total the training data contain 20 sample sequences from each component of length 200. The problem is non-trivial both because the data have exactly the same marginal statistics if the temporal sequence information is removed and because the Markov dynamics (as governed by $A_1$ and $A_2$) are relatively similar for each component making identification somewhat difficult.

The HMM clustering algorithm was applied to these sequences. The symmetrized likelihood distance matrix is shown as a grey-scale image in Figure 2. The axes have been ordered so that the sequences from the same clusters are adjacent. The difference in distances between the two clusters is apparent and the hierarchical clustering algorithm (with $K = 2$) easily separates the two groups. This initial clustering, followed by training separately the two clusters on the set of sequences assigned to each cluster, yielded:

$$\hat{A}_1 = \begin{pmatrix} 0.580 & 0.402 \\ 0.420 & 0.598 \end{pmatrix} \qquad \hat{\mu}_1 = \begin{pmatrix} 2.892 \\ 0.040 \end{pmatrix} \qquad \hat{\sigma}_1 = \begin{pmatrix} 1.353 \\ 1.219 \end{pmatrix}$$

$$\hat{A}_2 = \begin{pmatrix} 0.392 & 0.611 \\ 0.608 & 0.389 \end{pmatrix} \qquad \hat{\mu}_2 = \begin{pmatrix} 2.911 \\ 0.138 \end{pmatrix} \qquad \hat{\sigma}_2 = \begin{pmatrix} 1.239 \\ 1.339 \end{pmatrix}$$

Subsequent training of the composite model on all of the sequences produced more refined parameter estimates, although the basic cluster structure of the model remained the same (i.e., the initial clustering was robust).

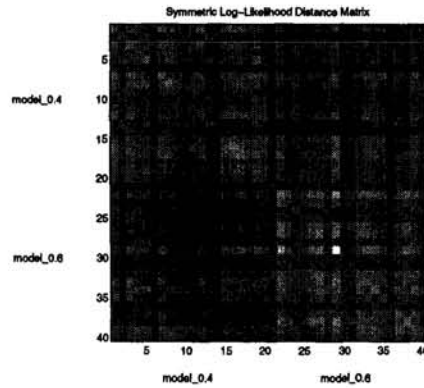

Figure 2: Symmetrized log-likelihood distance matrix.

For comparative purposes two alternative initialization procedures were used to initialize the training of the composite HMM. The "unstructured" method uniformly initializes the $A$ matrix without any knowledge of the fact that the off-block-diagonal terms are zero (this is the "standard" way of fitting a HMM). The "block-uniform" method uniformly initializes the $K$ block-diagonal matrices within $A$ and sets the off-block-diagonal terms to zero. Random initialization gave poorer results overall compared to uniform.

Table 1: Differences in log-likelihood for different initialization methods.

| Initialization Method | Maximum Log-Likelihood Value | Mean Log-Likelihood Value | Standard Deviation of Log-Likelihoods |
|---|---|---|---|
| Unstructured | 7.6 | 0.0 | 1.3 |
| Block-Uniform | 44.8 | 8.1 | 28.7 |
| HMM-Clustering | 55.1 | 50.4 | 0.9 |

The three alternatives were run 20 times on the data above, where for each run the seeds for the $k$-means component of the initialization were changed. The maximum, mean and standard deviations of log-likelihoods on test data are reported in Table 1 (the log-likelihoods were offset so that the mean unstructured log-likelihood is zero). The unstructured approach is substantially inferior to the others on this problem: this is not surprising since it is not given the block-diagonal structure of the true model. The Block-Uniform method is closer in performance to HMM-Clustering but is still inferior. In particular, its log-likelihood is consistently lower than that of the HMM-Clustering solution and has much greater variability across different initial seeds. The same qualitative behavior was observed across a variety of simulated data sets (results are not presented here due to lack of space).

## 3    Learning $K$, the Number of Sequence Components

### 3.1    Background

Above we have assumed that $K$, the number of clusters, is known. The problem of learning the "best" value for $K$ in a mixture model is a difficult one in practice

even for the simpler (non-dynamic) case of Gaussian mixtures. There has been considerable prior work on this problem. Penalized likelihood approaches are popular, where the log-likelihood on the training data is penalized by the subtraction of a complexity term. A more general approach is the full Bayesian solution where the posterior probability of each value of $K$ is calculated given the data, priors on the mixture parameters, and priors on $K$ itself. A potential difficulty here is the the computational complexity of integrating over the parameter space to get the posterior probabilities on $K$. Various analytic and sampling approximations are used in practice. In theory, the full Bayesian approach is fully optimal and probably the most useful. However, in practice the ideal Bayesian solution must be approximated and it is not always obvious how the approximation affects the quality of the final answer. Thus, there is room to explore alternative methods for determining $K$.

### 3.2 A Monte-Carlo Cross-Validation Approach

Imagine that we had a large test data set $D^{\text{test}}$ which is not used in fitting any of the models. Let $L_K(D^{\text{test}})$ be the log-likelihood where the model with $K$ components is fit to the training data $D$ but the likelihood is evaluated on $D^{\text{test}}$. We can view this likelihood as a function of the "parameter" $K$, keeping all other parameters and $D$ fixed. Intuitively, this "test likelihood" should be a much more useful estimator than the training data likelihood for comparing mixture models with different numbers of components. In fact, the test likelihood can be shown to be an unbiased estimator of the Kullback-Leibler distance between the true (but unknown) density and the model. Thus, maximizing out-of-sample likelihood over $K$ is a reasonable model selection strategy. In practice, one does not usually want to reserve a large fraction of one's data for test purposes: thus, a cross-validated estimate of log-likelihood can be used instead.

In Smyth (1996) it was found that for standard multivariate Gaussian mixture modeling, the standard $v$-fold cross-validation techniques (with say $v = 10$) performed poorly in terms of selecting the correct model on simulated data. Instead Monte-Carlo cross-validation (Shao, 1993) was found to be much more stable: the data are partitioned into a fraction $\beta$ for testing and $1 - \beta$ for training, and this procedure is repeated $M$ times where the partitions are randomly chosen on each run (i.e., need not be disjoint). In choosing $\beta$ one must tradeoff the variability of the performance estimate on the test set with the variability in model fitting on the training set. In general, as the total amount of data increases relative to the model complexity, the optimal $\beta$ becomes larger. For the mixture clustering problem $\beta = 0.5$ was found empirically to work well (Smyth, 1996) and is used in the results reported here.

### 3.3 Experimental Results

The same data set as described earlier was used where now $K$ is not known a priori. The 40 sequences were randomly partitioned 20 times into training and test cross-validation sets. For each train/test partition the value of $K$ was varied between 1 and 6, and for each value of $K$ the HMM-Clustering algorithm was fit to the training data, and the likelihood was evaluated on the test data. The mean cross-validated likelihood was evaluated as the average over the 20 runs. Assuming the models are equally likely a priori, one can generate an approximate posterior distribution $p(K|D)$ by Bayes rule: these posterior probabilities are shown in Figure 3. The cross-validation procedure produces a clear peak at $K = 2$ which is the true model size. In general, the cross-validation method has been tested on a variety of other simulated sequence clustering data sets and typically converges as a function of the number of training samples to the true value of $K$ (from below). As the number of

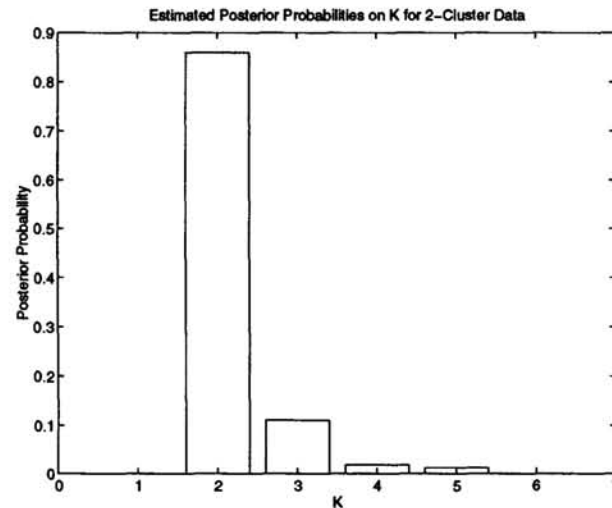

Figure 3: Posterior probability distribution on $K$ as estimated by cross-validation.

data points grow, the posterior distribution on $K$ narrows about the true value of $K$. If the data were not generated by the assumed form of the model, the posterior distribution on $K$ will tend to be peaked about the model size which is closest (in K-L distance) to the true model. Results in the context of Gaussian mixture clustering(Smyth 1996) have shown that the Monte Carlo cross-validation technique performs as well as the better Bayesian approximation methods and is more robust then penalized likelihood methods such as BIC.

In conclusion, we have shown that model-based probabilistic clustering can be generalized from feature-space clustering to sequence clustering. Log-likelihood between sequence models and sequences was found useful for detecting cluster structure and cross-validated likelihood was shown to be able to detect the true number of clusters.

## References

Baldi, P. and Y. Chauvin, 'Hierarchical hybrid modeling, HMM/NN architectures, and protein applications,' *Neural Computation*, 8(6), 1541–1565, 1996.

Krogh, A. et al., 'Hidden Markov models in computational biology: applications to protein modeling,' it J. Mol. Bio., 235:1501–1531, 1994.

Juang, B. H., and L. R. Rabiner, 'A probabilistic distance measure for hidden Markov models,' *AT&T Technical Journal*, vol.64, no.2, February 1985.

Rabiner, L. R., C. H. Lee, B. H. Juang, and J. G. Wilpon, 'HMM clustering for connected word recognition,' *Proc. Int. Conf. Ac. Speech. Sig. Proc*, IEEE Press, 405–408, 1989.

Shao, J., 'Linear model selection by cross-validation,' *J. Am. Stat. Assoc.*, 88(422), 486–494, 1993.

Smyth, P., 'Clustering using Monte-Carlo cross validation,' in *Proceedings of the Second International Conference on Knowledge Discovery and Data Mining*, Menlo Park, CA: AAAI Press, pp.126–133, 1996.

Zeevi, A. J., Meir, R., Adler, R., 'Time series prediction using mixtures of experts,' in this volume, 1997.
